# Modularity in the motor system: decomposition of muscle patterns as combinations of time-varying synergies

**Andrea d'Avella and Matthew C. Tresch**
Department of Brain and Cognitive Sciences
Massachusetts Institute of Technology, E25-526
Cambridge, MA 02139
{davel, mtresch}@ai.mit.edu

## Abstract

The question of whether the nervous system produces movement through the combination of a few discrete elements has long been central to the study of motor control. Muscle synergies, i.e. coordinated patterns of muscle activity, have been proposed as possible building blocks. Here we propose a model based on combinations of muscle synergies with a specific amplitude and temporal structure. Time-varying synergies provide a realistic basis for the decomposition of the complex patterns observed in natural behaviors. To extract time-varying synergies from simultaneous recording of EMG activity we developed an algorithm which extends existing non-negative matrix factorization techniques.

## 1 Introduction

In order to produce movement, every vertebrate has to coordinate the large number of degrees of freedom in the musculoskeletal apparatus. How this coordination is accomplished by the central nervous system is a long standing question in the study of motor control. According to one common proposal, this task might be simplified by a modular organization of the neural systems controlling movement [1, 2, 3, 4]. In this scheme, specific output modules would control different but overlapping sets of degrees of freedom, thereby decreasing the number of variables controlled by the nervous system. By activating different output modules simultaneously but independently, the system may achieve the flexibility necessary to control a variety of behaviors.

Several studies have sought evidence for such a modular controller by examining the patterns of muscle activity during movement, in particular looking for the presence of *muscle synergies*. A muscle synergy is a functional unit coordinating the activity of a number of muscles. The simplest model for such a unit would be the synchronous activation of a set of muscles with a specific activity balance, i.e. a vector in the muscle activity space. Using techniques such as the correlation between pairs of muscles, these studies have generally failed to provide strong evidence in support of such units. However, using a new analysis that allows for simultaneous combinations of more than one synergy, our group has recently provided evidence in support of this basic hypothesis of the neural control of movement.

We used a non-negative matrix factorization algorithm to examine the composition of muscle activation patterns in spinalized frogs [5, 6]. This algorithm, similarly to that developed independently by others [7], extracts a small number of non-negative[1] factors which can be combined to reconstruct a set of high-dimensional data.

However, this analysis assumed that the muscle synergies consisted of a set of muscles which were activated *synchronously*. In examinations of complex behaviors produced by intact animals, it became clear that muscles within a putative synergy were often activated asynchronously. In these cases, although the temporal delay between muscles was nonzero, the dispersion around this delay was very small. These observations suggested that the basic units of motor production might involve not only a fixed coordination of relative muscle activation amplitudes, but also a coordination of relative muscle activation timings. We therefore have developed a new algorithm to factorize muscle activation patterns produced during movement into combinations of such *time-varying muscle synergies*.

## 2  Combinations of time-varying muscle synergies

We model the output of the neural controller as a linear combination of $N$ muscle patterns with a specific time course in the activity of each muscle. In discrete time, we can represent each pattern, or *time-varying synergy*, as a sequence of vectors $\mathbf{w}(t)$ in muscle activity space. The data set which we consider here consists of episodes of a given behavior, e.g. a set of jumps in different directions and distances, or a set of walking or swimming cycles. In a particular episode $s$, each synergy is scaled by an amplitude coefficient $c_s$ and time-shifted by a delay $t_s$. The sequence of muscle activity for that episode is then given by:

$$\mathbf{m}_s(t) = \sum_{i=1}^{N} c_{si}\, \mathbf{w}_i(t - t_{si}) \tag{1}$$

Fig. 1 illustrates the model with an example of the construction of a muscle pattern by combinations of three synergies. Compared to the model based on combinations of *synchronous muscle synergies* this model has more parameters describing each synergy ($M \times T$ vs. $M$, with $M$ muscles and $T$ maximum number of time steps in a synergy) but less overall parameters. In fact, with synchronous synergies there is a combination coefficient for each time step and each synergy, whereas with time-varying synergies there are only two parameters ($c_{si}$ and $t_{si}$) for each episode and each synergy.

## 3  Iterative minimization of the reconstruction error

For a given set of episodes, we search for the set of $N$ non-negative time-varying synergies $\{\mathbf{W}_i\}_{i=1...N}$, $\mathbf{W}_i = [\mathbf{w}_i(0)...\mathbf{w}_i(T-1)]$, of maximum duration $T$ time steps and the set of coefficients $c_{is}(\geq 0)$ and $t_{is}$ that minimize the reconstruction error

$$E^2 = \sum_s E_s{}^2$$

$$E_s{}^2 = \sum_{t=1}^{T_s} \left\| \mathbf{m}_s(t) - \sum_{i=1}^{N} c_{si}\, \mathbf{w}_i(t - t_{si}) \right\|^2$$

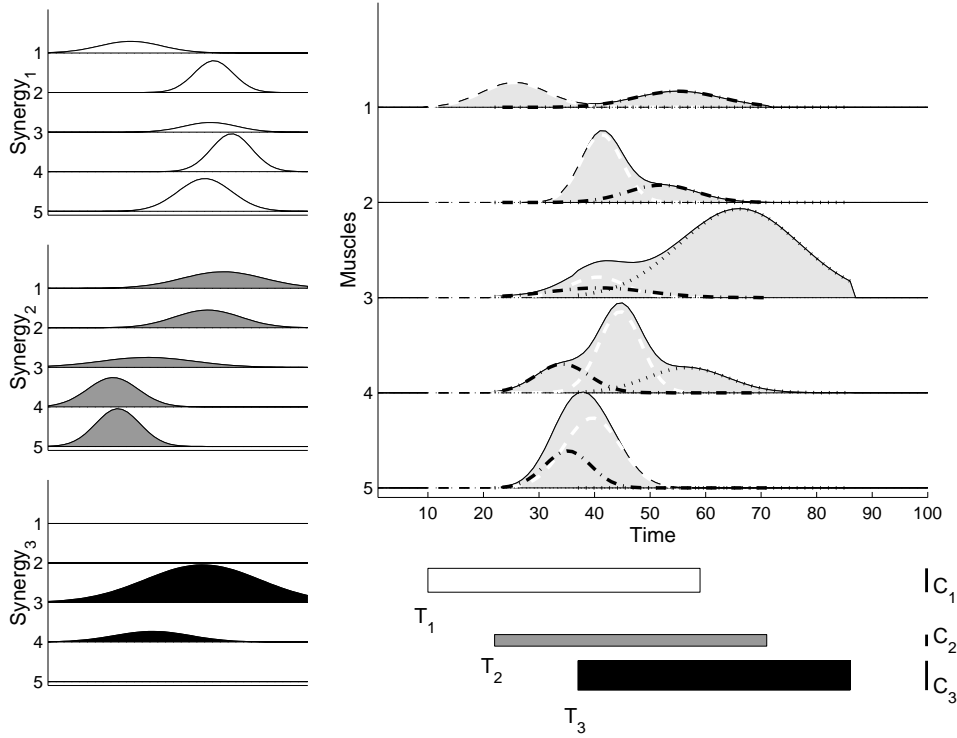

Figure 1: An example of construction of a muscle pattern by the combinations of three time-varying synergies. In this example, each time-varying synergy (*left*) is constituted by a sequence of 50 activation levels in 5 muscles chosen as samples from Gaussian functions with different centers, widths, and amplitudes. To construct the muscle pattern (*top right*, shaded area), the activity levels of each synergy are first scaled by an amplitude coefficient ($C_i$, represented in the *bottom right* by the height of an horizontal bar) and shifted in time by a delay ($T_i$, represented by the position of the same bar). Then, at each time step, the scaled and shifted components (*top right*, broken lines) are summed together.

with $\mathbf{w}_i(\tau) = 0$ for $\tau < 0$ or $\tau \geq T$.

After initializing synergies and coefficients to random positive values, we minimize the error by iterating the following steps:

1. For each episode, given the synergies $\mathbf{W}_i$ and the scaling coefficients $c_{si}$, find the delays $t_{si}$ using a nested matching procedure based on the cross-correlation of the synergies with the data (see 3.1 below).

2. For each episode, given the synergies and the delays $t_{si}$, update the scaling coefficients $c_{si}$ by gradient descent

$$\Delta \mathbf{c}_s = -\mu_C \nabla_{\mathbf{c}_s} E_s^2$$

Here and below, we enforce non-negativity by setting to zero any negative value.

3. Given delays and scaling coefficients, update the synergy elements $\mathbf{w}_{i\tau} = \mathbf{w}_i(\tau)$ by gradient descent

$$\Delta \mathbf{w}_{i\tau} = -\mu_W \nabla_{\mathbf{w}_{i\tau}} E^2$$

## 3.1 Matching the synergy delays

To find the best delay of each synergy in each episode we use the following procedure:

    i. Compute the sum of the scalar products between the $s$-th data episode and the $i$-th synergy time-shifted by $t$

$$\phi_{si}(t) = \sum_{\tau} \mathbf{m}_s(\tau)^T \mathbf{w}_i(\tau - t) \tag{2}$$

    or scalar product cross-correlation at delay $t$, for all possible delays.

    ii. Select the synergy and the delay with highest cross-correlation.

    iii. Subtract from the data the selected synergy (after scaling and time-shifting by the selected delay).

    iv. Repeat the procedure for the remaining synergies.

## 4 Results

We tested the algorithm on simulated data in order to evaluate its performance and then applied it to EMG recordings from 13 hindlimb muscles of intact bullfrogs during several episodes of natural behaviors [8].

### 4.1 Simulated data

We first tested whether the algorithm could reconstruct known synergies and coefficients from a dataset generated by those same synergies and coefficients. We used two different types of simulated synergies. The first type was generated using a Gaussian function of different center, width, and amplitude for each muscle. The second type consisted of synergies generated by uniformly distributed random activities. For each type, we generated sets of three synergies involving five muscles with a duration of 15 time steps. Using these synergies, 50 episodes of duration 30 time steps were generated by scaling each synergy $w_i(t)$ with random coefficients $c_{si}$ and shifting it by random delays $t_{si}$.

In figure 2 the results of a run with Gaussian synergies are shown. Using as a convergence criterion a change in $R^2$ of less than $10^{-5}$ for 20 iterations, after 474 iterations the solution had $R^2 = 0.9978$. Generating and reconstructed synergy activations are shown side by side on the *left*, in gray scale. Scatter plots of generating vs. reconstructed scaling coefficients and temporal delays are shown in the *center* and on the *right* respectively. Both synergies and coefficients were accurately reconstructed by the algorithm.

In table 1, a summary of the results from 10 runs with Gaussian and random synergies is presented. We used the maximum of the scalar product cross-correlation between two normalized synergies (see eq. 2) to characterize their similarity. We compared two sets of synergies by matching the pairs in each set with the highest similarity and computing the mean similarity ($S_W$) between these pairs. All the synergy sets that we reconstructed ($W_{rec}$) had a high similarity with the generating set ($W_{gen}$). We also compared the generating and reconstructed scaling coefficients $c_{si}$ using their correlation coefficient $r_c$, and delays $t_{si}$ by counting the number of delay coefficients that were reconstructed correctly after compensating for possible lags in the synergies ($N_{\Delta t=0}$). The match in scaling coefficients and delays was in general very good. Only in a few runs with Gaussian synergies were the data correctly reconstructed (high $R^2$) but with synergies slightly different from the generating ones (as indicated by the lower $S_W$) and consequently not perfectly matching coefficients (lower $\langle r_c \rangle$ and $\langle N_{\Delta t=0}/N_t \rangle$).

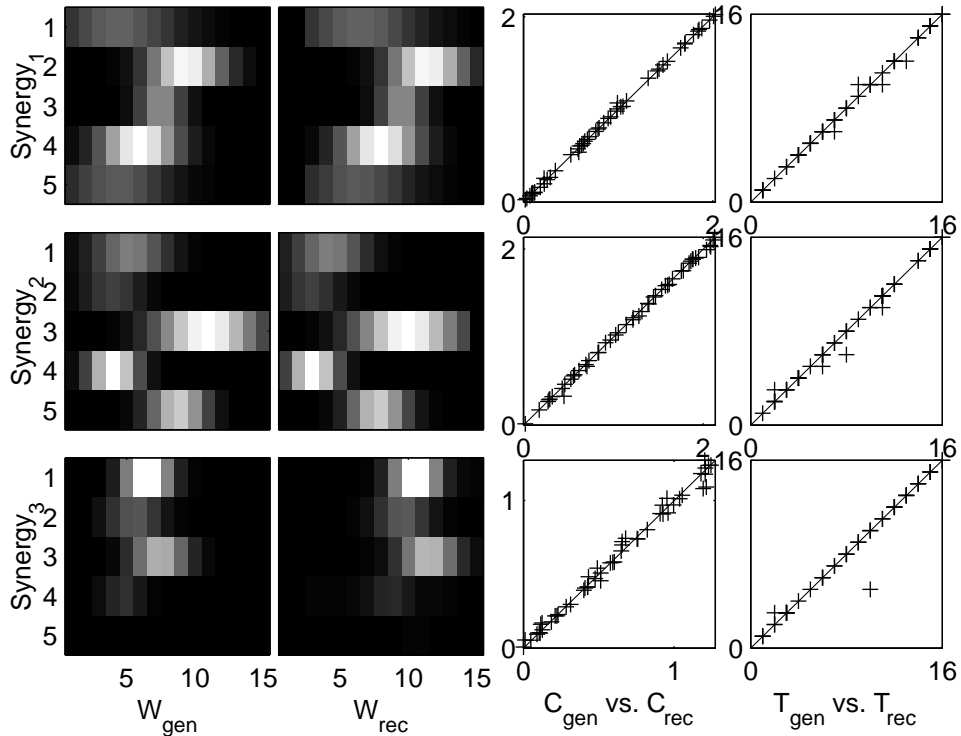

Figure 2: An example of reconstruction of known synergies and coefficients from simulated data. The *first column* ($W_{gen}$) shows three time-varying synergies, generated from Gaussian functions, as three matrices each representing, in gray scale, the activity of 5 muscles (*rows*) over 15 time steps (*columns*). The *second column* ($W_{rec}$) shows the three synergies reconstructed by the algorithm: they accurately match the generating synergies (except for a temporal shift compensated by an opposite shift in the reconstructed delays). The *third and fourth columns* show scatter plots of generating vs. reconstructed scaling coefficients and delays in 50 simulated episodes. Both sets of coefficients are accurately reconstructed in almost all episodes.

## 4.2 Time-varying muscle synergies in frog's muscle patterns

We then applied the algorithm to EMG recordings of a large set ($n = 111$) of hindlimb kicks, a defensive reflex that frogs use to remove noxious stimuli from the foot. Each kick consists of a fast extension followed by a slower flexion to return the leg to a crouched posture. The trajectory of the foot varies with the location of the stimulation on the skin and, as a consequence, the set of kicks spans a wide range of the workspace of the frog. Correspondingly, across different episodes the muscle activity patterns in the 13 muscles that we recorded showed considerable amplitude and timing variations that we sought to explain by combinations of time-varying synergies.

After rectifying and integrating the EMGs over 10 ms intervals, we performed the optimization procedure with sets of $N$ synergies, with $N = 2, ..., 5$. We chose the maximum duration of each synergy to be 20 time steps, i.e. 200 ms, a duration larger than the duration of a typical muscle burst observed in this behavior. We repeated the procedure 10 times for each $N$.

| Gaussian synergies | | | | | |
|---|---|---|---|---|---|
| | $N_{iter}$ | $R^2$ | $S_W$ | $\langle r_c \rangle$ | $\langle N_{\Delta t = 0}/N_t \rangle$ |
| max | 561 | 0.9989 | 0.9996 | 0.9983 | 0.9467 |
| median | 451 | 0.9952 | 0.9990 | 0.9974 | 0.9233 |
| min | 297 | 0.9874 | 0.8338 | 0.2591 | 0.3133 |
| Random synergies | | | | | |
| | $N_{iter}$ | $R^2$ | $S_W$ | $\langle r_c \rangle$ | $\langle N_{\Delta t = 0}/N_t \rangle$ |
| max | 555 | 0.9999 | 1.0000 | 0.9996 | 0.9867 |
| median | 395 | 0.9998 | 1.0000 | 0.9990 | 0.9800 |
| min | 208 | 0.9998 | 1.0000 | 0.9984 | 0.9733 |

Table 1: Comparison between generated and reconstructed synergies and coefficients for 10 runs with Gaussian and random synergies. See text for explanation.

In figure 3 the result of the extraction of four synergies with the highest $R^2$ is shown. The convergence criterion of a change in $R^2$ smaller than $10^{-4}$ for 20 iterations was reached after 100 iterations with a final $R^2 = 0.6685$. The synergies extracted in the other nine runs were in general very similar to this set, as indicated by a mean similarity ($S_W$) ranging from $0.99$ to $0.87$ (median $0.96$) and a correlation between scaling coefficients ranging from $0.99$ to $0.59$ (median $0.96$). In the case with the lowest similarity, only one synergy in the set shown in figure 3 was not properly matched.

The four synergies captured the basic features of the muscle patterns observed during different kicks. The first synergy, recruiting all the major knee extensor muscles (VI, RA, and VE), is highly activated in laterally directed kicks, as seen in the first kick shown in figure 3, which involved a large knee extension. The second synergy, recruiting two large hip extensor muscles (RI and SM) and an ankle extensor muscle (GA), is highly activated in caudally and medially directed kicks, i.e. kicks involving hip extension. The third synergy involves a specific temporal sequencing of several muscles: BI and VE first, followed by RI, SM, and GA, and then by AD and VI at the end. The fourth synergy has long activation profiles in many flexor muscles, i.e. those involved in the return phase of the kick, with a specific temporal pattern (long activation of IP; BI and SA before TA).

When this set of EMGs was reconstructed using different numbers of muscle synergies, we found that the synergies identified using N synergies were generally preserved in the synergies identified using N+1 synergies. For instance, the first two synergies shown in figure 3 were seen in all sets of synergies, from $N = 2$ to $N = 5$. Therefore, increasing the number of synergies allowed the data to be reconstructed more accurately (as seen by a higher $R^2$) but without a complete reorganization of the synergies.

## 5 Discussion

The algorithm that we introduced here represents a new analytical tool for the investigation of the organization of the motor system. This algorithm is an extension of previous non-negative matrix factorization procedures, providing a means of capturing structure in a set of data not only in the amplitude domain but also in the temporal domain. Such temporal structure is a natural description of motor systems where many behaviors are characterized by a particular temporal organization. The analysis applied to behaviors produced by the frog, as described here, was able to capture significant physiologically relevant characteristics in the patterns of muscle activations. The motor system is not unique, however, in having structure in both amplitude and temporal domains and the techniques used here could easily be extended to other systems.

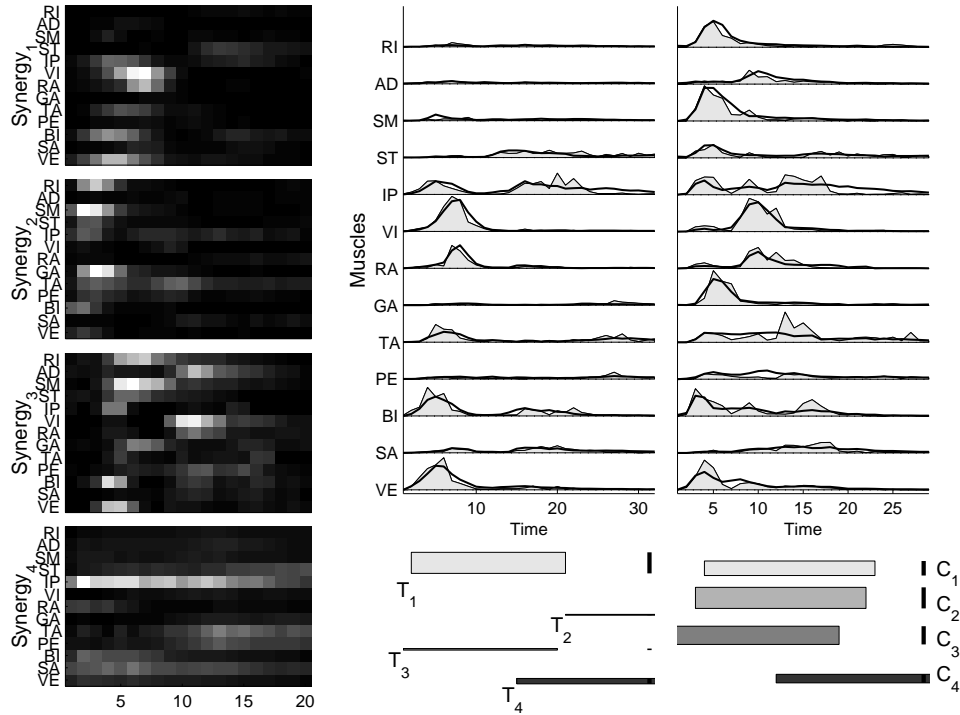

Figure 3: Reconstruction of rectified and integrated (10 ms) EMGs for two kicks by time-varying synergies. *Left*: four extracted synergies constituted by activity levels (in gray scale) for 20 time steps in 13 muscles: rectus internus major (RI), adductor magnus (AD), semimembranosus (SM), ventral head of semitendinosus (ST), ilio-psoas (IP), vastus internus (VI), rectus anterior (RA), gastrocnemius (GA), tibialis anterior (TA), peroneous (PE), biceps (BI), sartorius (SA), and vastus externus (VE) [8]. *Top right*: the observed EMGs (*thin line and shaded area*) and their reconstruction (*thick line*) by combinations of the four synergies, scaled in amplitude ($C_i$) and shifted in time ($T_i$).

Our model can be naturally extended to include temporal scaling of the synergies, i.e. allowing different durations of a synergy in different episodes. Work is in progress to implement an algorithm similar to the one presented here to extract time-varying and time-scalable synergies. We will also address the issue of how to identify time-varying muscle synergies from continuous recordings of EMG patterns, without any manual segmentation into different episodes. A possibility that we are investigating is to extend the approach based on a sparse and overcomplete basis used by Lewicki and Sejnowski [9]. Finally, future work will aim to the development of a probabilistic model to address the issue of the dimensionality of the synergy set in terms of Bayesian model selection [10].

**Acknowledgments**

We thank Zoubin Ghahramani, Emanuel Todorov, Emilio Bizzi, Sebastian Seung, Simon Overduin, and Maura Mezzetti for useful discussions and comments.

## Footnotes

[1]The non-negativity constraint arises naturally in the context of motor control from the fact that firing rates of motoneurons, and consequently muscle activities, cannot be negative. While it is conceivable that a negative contribution on a motoneuronal pool from one factor would always be cancelled by a larger positive contribution from other factors, we chose a model based on non-negative factors to ensure that each factor could be independently activated.

# References

[1] E. Bizzi, P. Saltiel, and M. Tresch. Modular organization of motor behavior. *Z Naturforsch [C]*, 53(7-8):510–7, 1998.

[2] F. A. Mussa-Ivaldi. Modular features of motor control and learning. *Curr Opin Neurobiol*, 9(6):713–7, 1999.

[3] W. J. Kargo and S. F. Giszter. Rapid correction of aimed movements by summation of force-field primitives. *J Neurosci*, 20(1):409–26, 2000.

[4] Z. Ghahramani and D. M. Wolpert. Modular decomposition in visuomotor learning. *Nature*, 386(6623):392–5, 1997.

[5] M. C. Tresch, P. Saltiel, and E. Bizzi. The construction of movement by the spinal cord. *Nature Neuroscience*, 2(2):162–7, 1999.

[6] P. Saltiel, K. Wyler-Duda, A. d'Avella, M. C. Tresch, and E. Bizzi. Muscle synergies encoded within the spinal cord: evidence from focal intraspinal nmda iontophoresis in the frog. *Journal of Neurophysiology*, 85(2):605–19, 2001.

[7] D. D. Lee and H. S. Seung. Learning the parts of objects by non-negative matrix factorization. *Nature*, 401(6755):788–91, 1999.

[8] A. d'Avella. *Modular control of natural motor behavior*. PhD thesis, MIT, 2000.

[9] M. S. Lewicki and T. J. Sejnowski. Coding time-varying signals using sparse, shift-invariant representations. In M. S. Kearns, S. A. Solla, and D. A. Cohn, editors, *Advances in Neural Information Processing Systems 11*. MIT Press, 1999.

[10] L. Wasserman. Bayesian model selection and model averaging. *Journal of Mathematical Psychology*, 44:92–107, 2000.
